# The Observer-Observation Dilemma in Neuro-Forecasting

**Hans Georg Zimmermann**
Siemens AG
Corporate Technology
D-81730 München, Germany
Georg.Zimmermann@mchp.siemens.de

**Ralph Neuneier**
Siemens AG
Corporate Technology
D-81730 München, Germany
Ralph.Neuneier@mchp.siemens.de

## Abstract

We explain how the training data can be separated into clean information and unexplainable noise. Analogous to the data, the neural network is separated into a time invariant structure used for forecasting, and a noisy part. We propose a unified theory connecting the optimization algorithms for cleaning and learning together with algorithms that control the data noise and the parameter noise. The combined algorithm allows a data-driven local control of the liability of the network parameters and therefore an improvement in generalization. The approach is proven to be very useful at the task of forecasting the German bond market.

## 1 Introduction: The Observer-Observation Dilemma

Human beings believe that they are able to solve a psychological version of the *Observer-Observation Dilemma*. On the one hand, they use their observations to constitute an understanding of the laws of the world, on the other hand, they use this understanding to evaluate the correctness of the incoming pieces of information. Of course, as everybody knows, human beings are not free from making mistakes in this psychological dilemma. We encounter a similar situation when we try to build a mathematical model using data. Learning relationships from the data is only one part of the model building process. Overrating this part often leads to the phenomenon of overfitting in many applications (especially in economic forecasting). In practice, evaluation of the data is often done by external knowledge, i. e. by optimizing the model under constraints of smoothness and regularization [7]. If we assume, that our model summerizes the best knowledge of the system to be identified, why should we not use the model itself to evaluate the correctness of the data? One approach to do this is called Clearning [11]. In this paper, we present a unified approach of the interaction between the data and a neural network (see also [8]). It includes a new symmetric view on the optimization algorithms, here learning and cleaning, and their control by parameter and data noise.

## 2 Learning

### 2.1 Learning reviewed

We are especially interested in using the output of a neural network $y(x, w)$, given the input pattern, $x$, and the weight vector, $w$, as a forecast of financial time series. In the context of neural networks learning normally means the minimization of an error function $E$ by changing the weight vector $w$ in order to achieve good generalization performance. Typical error functions can be written as a sum of individual terms over all $T$ training patterns, $E = \frac{1}{T} \sum_{t=1}^{T} E_t$. For example, the maximum-likelihood principle leads to

$$E_t = 1/2 \left( y(x, w) - y_t^d \right)^2, \tag{1}$$

with $y_t^d$ as the given target pattern. If the error function is a nonlinear function of the parameters, learning has to be done iteratively by a search through the weight space, changing the weights from step $\tau$ to $\tau + 1$ according to:

$$w^{(\tau+1)} = w^{(\tau)} + \Delta w^{(\tau)}. \tag{2}$$

There are several algorithms for choosing the weight increment $\Delta w^{(\tau)}$, the most easiest being *gradient descent*. After each presentation of an input pattern, the gradient $g_t := \nabla E_t|_w$ of the error function with respect to the weights is computed. In the batch version of gradient descent the increments are based on all training patterns

$$\Delta w^{(\tau)} = -\eta g = -\eta \frac{1}{T} \sum_{t=1}^{T} g_t, \tag{3}$$

whereas the pattern-by-pattern version changes the weights after each presentation of a pattern $x_t$ (often randomly chosen from the training set):

$$\Delta w^{(\tau)} = -\eta g_t. \tag{4}$$

The learning rate $\eta$ is typically held constant or follows an annealing procedure during training to assure convergence. Our experiments have shown that small batches are most useful, especially in combination with Vario-Eta, a stochastic approximation of a Quasi-Newton method [3]:

$$\Delta w^{(\tau)} = -\frac{\eta}{\sqrt{\frac{1}{T} \sum (g_t - g)^2}} \cdot \frac{1}{N} \sum_{t=1}^{N} g_t, \tag{5}$$

with and $N \leq 20$. Learning pattern-by-pattern or with small batches can be viewed as a stochastic search process because we can write the weight increments as:

$$\Delta w^{(\tau)} = -\eta \left[ g + \left( \frac{1}{N} \sum_{t=1}^{N} g_t - g \right) \right]. \tag{6}$$

These increments consist of the terms $g$ with a drift to a local minimum and of noise terms $(\frac{1}{N} \sum_{t=1}^{N} g_t - g)$ disturbing this drift.

### 2.2 Parameter Noise as an Implicit Penalty Function

Consider the Taylor expansion of $E(w)$ around some point $w$ in the weight space

$$E(w + \Delta w) = E(w) + \nabla E \, \Delta w + \frac{1}{2} \Delta w' H \Delta w \tag{7}$$

with $H$ as the Hessian of the error function. Assume a given sequence of $T$ disturbance vectors $\Delta w_t$, whose elements $\Delta w_t(i)$ are identically, independently distributed (i.i.d.) with zero mean and variance (row-)vector $\mathrm{var}(\Delta w_i)$ to approximate the expectation $\langle E(w) \rangle$ by

$$\langle E(w) \rangle \approx \frac{1}{T} \sum_t E(w + \Delta w_t) = E(w) + \frac{1}{2} \sum_i \mathrm{var}(\Delta w(i)) H_{ii}, \qquad (8)$$

with $H_{ii}$ as the diagonal elements of $H$. In eq. 8, noise on the weights acts implicitly as a penalty term to the error function given by the second derivatives $H_{ii}$. The noise variances $\mathrm{var}(\Delta w(i))$ operate as penalty parameters. As a result of this flat minima solutions which may be important for achieving good generalization performance are favored [5].

Learning pattern-by-pattern introduces such noise in the training procedure i.e., $\Delta w_t = -\eta \cdot g_t$. Close to convergence, we can assume that $g_t$ is i.i.d. with zero mean and variance vector $\mathrm{var}(g_i)$ so that the expected value can be approximated by

$$\langle E(w) \rangle \approx E(w) + \frac{\eta^2}{2} \sum_i \mathrm{var}(g_i) \frac{\partial^2 E}{\partial w_i^2}. \qquad (9)$$

This type of learning introduces to a local penalty parameter $\mathrm{var}(\Delta w(i))$, characterizing the stability of the weights $w = [w_i]_{i=1,\ldots,k}$.

The noise effects due to Vario-Eta learning $\Delta w_t(i) = -\frac{\eta}{\sqrt{\sigma_i^2}} \cdot g_{ti}$ leads to

$$\langle E(w) \rangle \approx E(w) + \frac{\eta^2}{2} \sum_i \frac{\partial^2 E}{\partial w_i^2}. \qquad (10)$$

By canceling the term $\mathrm{var}(g_i)$ in eq. 9, Vario-Eta achieves a simplified uniform penalty parameter, which depends only on the learning rate $\eta$. Whereas pattern-by-pattern learning is a slow algorithm with a locally adjusted penalty control, Vario-Eta is fast only at the cost of a simplified uniform penalty term. We summarize this section by giving some advice on how to learn to flat minima solutions:

- Train the network to a minimal training error solution with Vario-Eta, which is a stochastic approximation of a Newton method and therefore very fast.

- Add a final phase of pattern-by-pattern learning with uniform learning rate to fine tune the local curvature structure by the local penalty parameters (eq. 9).

- Use a learning rate $\eta$ as high as possible to keep the penalty effective. The training error may vary a bit, but the inclusion of the implicit penalty is more important.

## 3  Cleaning

### 3.1  Cleaning reviewed

When training neural networks, one typically assumes that the data is noise-free and one forces the network to fit the data exactly. Even the control procedures to minimize over-fitting effects (i.e., pruning) consider the inputs as exact values. However, this assumption is often violated, especially in the field of financial analysis, and we are taught by the phenomenon of overfitting not to follow the data exactly. Clearning, as a combination of cleaning and learning, has been introduced in the paper of [11]. The motivation was to minimize overfitting effects by considering the input data as corrupted by noise whose distribution has also to be learned. The Cleaning error function for the pattern $t$ is given by the sum of two terms

$$E_t^{y,x} = \frac{1}{2}\left[ \left(y_t - y_t^d\right)^2 + \left(x_t - x_t^d\right)^2 \right] = E_t^y + E_t^x \qquad (11)$$

with $x_t^d, y_t^d$ as the observed data point. In the pattern-by-pattern learning, the network output $y(x_t, w)$ determines the adaptation as usual,

$$w^{(\tau+1)} = w^{(\tau)} - \eta \frac{\partial E^y}{\partial w^{(\tau)}}. \tag{12}$$

We have also to memorize correction vectors $\Delta x_t$ for all input data of the training set to present the cleaned input $x_t$ to the network,

$$x_t = x_t^d + \Delta x_t \tag{13}$$

The update rule for the corrections, initialized with $\Delta x_t^{(0)} = 0$ can be described as

$$\Delta x_t^{(\tau+1)} = (1-\eta)\Delta x_t^{(\tau)} - \eta(y_t - y_t^d)\frac{\partial y}{\partial x} \tag{14}$$

All the necessary quantities, i. e. $(y_t - y_t^d)\frac{\partial y(x,w)}{\partial x}$ are computed by typical back-propagation algorithms, anyway. We experienced, that the algorithms work well, if the same learning rate $\eta$ is used for both, the weight and cleaning updates. For regression, cleaning forces the acceptance of a small error in $x$, which can in turn decrease the error in $y$ dramatically, especially in the case of outliers. Successful applications of Cleaning are reported in [11] and [9].

Although the network may learn an optimal model for the cleaned input data, there is no easy way to work with cleaned data on the test set. As a consequence, the model is evaluated on a test set with a different noise characteristic compared to the training set. We will later propose a combination of learning with noise and cleaning to work around this serious disadvantage.

## 3.2 Data Noise reviewed

Artificial noise on the input data is often used during training because it creates an infinite number of training examples and expands the data to empty parts of the input space. As a result, the tendency of learning by heart may be limited because smoother regression functions are produced.

Now, we are considering again the Taylor expansion, this time applied to $E(x)$ around some point $x$ in the input space. The expected value $\langle E(x) \rangle$ is approximated by

$$\langle E(x) \rangle \approx \frac{1}{T} \sum_t E(x + \Delta x_t) = E(x) + \frac{1}{2} \sum_j \text{var}(\Delta x(j)) H_{jj}, \tag{15}$$

with $H_{jj}$ as the diagonal elements of the Hessian $H_{xx}$ of the error function with respect to the inputs $x$. Again, in eq. 15, noise on the inputs acts implicitly as a penalty term to the error function with the noise variances $\text{var}(\Delta x(j))$ operating as penalty parameters. Noise on the input improve generalization behavior by favoring smooth models [1].

The noise levels can be set to a constant value, e. g. given by a priori knowledge, or adaptive as described now. We will concentrate on a uniform or normal noise distribution. Then, the adaptive noise level $\xi_j$ is estimated for each input $j$ individually. Suppressing pattern indices, we define the noise levels $\xi_j$ or $\xi_j^2$ as the average residual errors:

$$\text{uniform residual error:} \quad \xi_j = \frac{1}{T} \sum_t \left| \frac{\partial E^y}{\partial x_j} \right|, \tag{16}$$

$$\text{Gaussian residual error:} \quad \xi_j^2 = \frac{1}{T} \sum_t \left( \frac{\partial E^y}{\partial x_j} \right)^2. \tag{17}$$

Actual implementations use stochastic approximation, e. g. for the uniform residual error

$$\xi_j^{(\tau+1)} = (1 - \frac{1}{T})\xi_j^{(\tau)} + \frac{1}{T} \left| \frac{\partial E^y}{\partial x_j} \right|. \tag{18}$$

The different residual error levels can be interpreted as follows: A small level $\xi_j$ may indicate an unimportant input $j$ or a perfect fit of the network concerning this input $j$. In both cases, a small noise level is appropriate. On the other hand, a high value of $\xi_j$ for an input $j$ indicates an important but imperfectly fitted input. In this case high noise levels are advisable. High values of $\xi_j$ lead to a stiffer regression model and may therefore increase the generalization performance of the network.

### 3.3 Cleaning with Noise

Typically, training with noisy inputs takes a data point and adds a random variable drawn from a fixed or adaptive distribution. This new data point $x_t$ is used as an input to the network. If we assume, that the data is corrupted by outliers and other influences, it is preferable to add the noise term to the cleaned input. For the case of Gaussian noise the resulting new input is:

$$x_t = x_t^d + \Delta x_t + \xi \phi, \tag{19}$$

with $\phi$ drawn from the normal distribution. The cleaning of the data leads to a corrected mean of the data and therefore to a more symmetric noise distribution, which also covers the observed data $x_t$.

We propose a variant which allows more complicated noise distributions:

$$x_t = x_t^d + \Delta x_t - \Delta x_k, \tag{20}$$

with $k$ as a random number drawn from the indices of the correction vectors $[\Delta x_t]_{t=1,...,T}$. In this way we use a possibly asymmetric and/or dependent noise distribution, which still covers the observed data $x_t$ by definition of the algorithm.

One might wonder, why to disturb the cleaned input $x_t^d + \Delta x_t$ with an additional noisy term $\Delta x_k$. The reason for this is, that we want to benefit from representing the whole input distribution to the network instead of only using one particular realization.

## 4   A Unifying Approach

### 4.1   The Separation of Structure and Noise

In the previous sections we explained how the data can be separated into clean information and unexplainable noise. Analogous, the neural network is described as a time invariant structure (otherwise no forecasting is possible) and a noisy part.

> *data*                      → *cleaned data*                *+time invariant data noise*
> *neural network* → *time invariant parameters* + *parameter noise*

We propose to use cleaning and adaptive noise to separate the data and to use learning and stochastic search to separate the structure of the neural network.

> *data*                  ← *cleaning(neural network)* + *adaptive noise (neural network)*
> *neural network* ← *learning (data)*                *+ stochastic search(data)*

The algorithms analyzing the data depend directly on the network whereas the methods searching for structure are directly related to the data. It should be clear that the model building process should combine both aspects in an alternate or simultaneous manner. The interaction of algorithms concerning data analysis and network structure enables the realization of the the concept of the Observer-Observation Dilemma.

The aim of the unified approach can be described, exemplary assuming here a Gaussian noise model, as the minimization of the error due to both, the structure and the data:

$$\frac{1}{2T}\sum_{t=1}^{T}\left[\left(y_t-y_t^d\right)^2+\left(x_t-x_t^d\right)^2\right]\to\min_{x_t,w} \quad (21)$$

Combining the algorithms and approximating the cumulative gradient $g$ by $\tilde{g}$, we receive

$$\overline{\begin{array}{rcl} \text{data} \\ \Delta x_t^{(\tau+1)} &=& (1-\eta)\Delta x_t^{(\tau)}-\eta(y_t-y_t^d)\frac{\partial y}{\partial x} \\ x_t &=& x_t^d+\underbrace{\Delta x_t^{(\tau)}}_{\text{cleaning}}-\underbrace{\Delta x_k^{(\tau)}}_{\text{noise}} \end{array}}$$

$$\overline{\begin{array}{rcl} \text{structure} \\ \tilde{g}^{(\tau+1)} &=& (1-\alpha)\tilde{g}^{(\tau)}+\alpha(y_t-y_t^d)\frac{\partial y}{\partial w} \\ w^{(\tau+1)} &=& w^{(\tau)}-\underbrace{\eta\tilde{g}^{(\tau)}}_{\text{learning}}-\underbrace{\eta(g_t-\tilde{g}^{(\tau)})}_{\text{noise}} \end{array}} \quad (22)$$

The cleaning of the data by the network computes an individual correction term for each training pattern. The adaptive noise procedure according to eq. 20 generates a potentially asymmetric and dependent noise distribution which also covers the observed data. The implied curvature penalty, whose strength depends on the individual liability of the input variables, can improve the generalization performance of the neural network.

The learning of the structure searches for time invariant parameters characterized by $\frac{1}{T}\sum g_t=0$. The parameter noise supports this exploration as a stochastic search to find better "global" minima. Additionally, the generalization performance may be further improved by the implied curvature penalty depending on the local liability of the parameters. Note that, although the description of the weight updates collapses to the simple form of eq. 4, we preferred the formula above to emphasize the analogy between the mechanism which handles the data and the structure.

In searching for an optimal combination of data and parameters, the noise of both parts is not a disastrous failure to build a perfect model but it is an important element to control the interaction of data and structure.

## 4.2 Pruning

The neural network topology represents only a hypothesis of the true underlying class of functions. Due to possible misspecification, we may have defects of the parameter noise distribution. Pruning algorithms are not only a way to limit the memory of the network, but they also appear useful to correct the noise distribution in different ways.

*Stochastic-Pruning* [2] is basically a t-test on the weights $w$. Weights with low test$_w$ values constitute candidates for pruning to cancel weights with low liability measured by the size of the weight divided by the standard deviation of its fluctuations. By this, we get a stabilization of the learning against resampling of the training data. A further weight pruning method is EBD, *Early-Brain-Damage* [10], which is based on the often cited OBD pruning method of [6]. In contrast to OBD, EBD allows its application before the training has reached a local minimum. One of the advantages of EBD over OBD is the possibility to perform the testing while being slidely away from a local minimum. In our training procedure we propose to use noise even in the final part of learning and therefore we are only nearby a local minimum. Furthermore, EBD is also able to revive already pruned weights. Similar to Stochastic Pruning, EBD favors weights with a low rate of fluctuations. If a weight is pushed around by a high noise, the implicit curvature penalty would favor a flat minimum around this weight which leads to its elimination by EBD.

## 5 Experiments

In a research project sponsored by the European Community we are applying the proposed approach to estimate the returns of 3 financial markets for each of the G7 countries subsequently using these estimations in an asset allocation scheme to create a Markowitz-optimal portfolio [4]. This paper reports the 6 month forecasts of the German bond rate, which is one of the more difficult tasks due to the reunification of Germany and GDR. The inputs consist of 39 variables achieved by preprocessing 16 relevant financial time series. The training set covers the time from April, 1974 to December, 1991, the test set runs from January, 1992 to May, 1996. The network arcitecture consists of one hidden layer (20 neurons, tanh transfer function) and one linear output. First, we trained the neural network until convergence with pattern-by-pattern learning using a small batch size of 20 patterns (classical approach). Then, we trained the network using the unified approach as described in section 4.1 using pattern-by-pattern learning. We compare the resulting predictions of the networks on the basis of four performance measures (see table). First, the hit rate counts how often the sign of the return of the bond has been correctly predicted. As to the other measures, the step from the forecast model to a trading system is here kept very simple. If the output is positive, we buy shares of the bond, otherwise we sell them. The potential realized is the ratio of the return to the maximum possible return over the test (training) set. The annualized return is the average yearly profit of the trading systems. Our approach turns out to be superior: we almost doubled the annualized return from 4.5% to 8.5% on the test set. The figure compares the accumulated return of the two approaches on the test set. The unified approach not only shows a higher profitability, but also has by far a less maximal draw down.

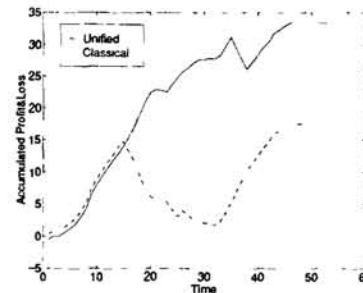

| approach | our | classical |
|---|---|---|
| hit rate | 81% (96%) | 66% (93%) |
| realized potential | 75% (100%) | 44% (96%) |
| annualized return | 8.5% (11.2%) | 4.5% (10.1%) |

## References

[1] Christopher M. Bishop. *Neural Networks for Pattern Recognition*. Clarendon Press, 1994.

[2] W. Finnoff, F. Hergert, and H. G. Zimmermann. Improving generalization performance by nonconvergent model selection methods. In proc. of *ICANN-92*, 1992.

[3] W. Finnoff, F. Hergert, and H. G. Zimmermann. Neuronale Lernverfahren mit variabler Schrittweite. 1993. Tech. report, Siemens AG.

[4] P. Herve, P. Naim, and H. G. Zimmermann. Advanced Adaptive Architectures for Asset Allocation: A Trial Application. In *Forecasting Financial Markets*, 1996.

[5] S. Hochreiter and J. Schmidhuber. Flat minima. *Neural Computation*, 9(1):1–42, 1997.

[6] Y. le Cun, J. S. Denker, and S. A. Solla. Optimal brain damage. *NIPS*89*, 1990.

[7] J. E. Moody and T. S. Rögnvaldsson. Smoothing regularizers for projective basis function networks. *NIPS 9*, 1997.

[8] R. Neuneier and H. G. Zimmermann. How to Train Neural Networks. In *Tricks of the Trade: How to make algorithms really to work*. Springer Verlag, Berlin, 1998.

[9] B. Tang, W. Hsieh, and F. Tangang. Clearning neural networks with continuity constraints for prediction of noisy time series. *ICONIP'96*, 1996.

[10] V. Tresp, R. Neuneier, and H. G. Zimmermann. Early brain damage. *NIPS 9*, 1997.

[11] A. S. Weigend, H. G. Zimmermann, and R. Neuneier. Clearning. *Neural Networks in Financial Engineering, (NNCM95)*, 1995.